# Mixture Regression for Covariate Shift

**Amos J Storkey**
Institute of Adaptive and Neural Computation
School of Informatics, University of Edinburgh
*a.storkey@ed.ac.uk*

**Masashi Sugiyama**
Department of Computer Science
Tokyo Institute of Technology
*sugi@cs.titech.ac.jp*

## Abstract

In supervised learning there is a typical presumption that the training and test points are taken from the same distribution. In practice this assumption is commonly violated. The situations where the training and test data are from different distributions is called *covariate shift*. Recent work has examined techniques for dealing with covariate shift in terms of minimisation of generalisation error. As yet the literature lacks a Bayesian generative perspective on this problem. This paper tackles this issue for regression models. Recent work on covariate shift can be understood in terms of mixture regression. Using this view, we obtain a general approach to regression under covariate shift, which reproduces previous work as a special case. The main advantages of this new formulation over previous models for covariate shift are that we no longer need to presume the test and training densities are known, the regression and density estimation are combined into a single procedure, and previous methods are reproduced as special cases of this procedure, shedding light on the implicit assumptions the methods are making.

## 1 Introduction

There is a common presumption in developing supervised methods that the distribution of training points used for learning supervised models will match the distribution of points seen in a new test scenario. The expectation that the training and test points follow the same distribution is explicitly stated in [2, p. 10], is an assumption of empirical risk minimisation, [see e.g. 9, p. 25], and is implicit in the common practice of randomized splitting given data into a "training set" and a "test set", where the latter is used in assessing performance [5, p. 482-495].

This paper, then, is concerned with the following issue. A set of real valued training data pairs of the form $(\mathbf{x}, \mathbf{y})$ is provided to train a model for a supervised learning problem. In addition data of the form $\mathbf{x}$ is provided from one (or more) test environments where the model will be used. The question to be addressed is "How should we predict a value of $\mathbf{y}$ given a value $\mathbf{x}$ from within that particular test environment?"

Cases where test scenarios truly match the training data are probably rare. The problem of mismatch has been grappled within literature from a number of fields, and has become known as *covariate shift* [14]. Specific examples of covariate shift include situations in reinforcement learning [c.f. 13] and bio-informatics [c.f. 1]. The common issue of sample selection bias [7] is a particular case of covariate shift.

Much of the recent analysis of covariate shift has been made in the context of assessing the asymptotic bias of various estimators [15]. In general it has been noted that in the case of mismatched models (i.e. where the model from which the training data is generated is not included in the training model class), some typical estimators, such as least squares approaches, produce biased asymptotic estimators [14]. It might appear that the presumption of matched models in Bayesian analysis means covariate shift is not an issue: failure or otherwise under situations of covariate shift is solved by valid choice for the prior distribution over conditional models. The difficulty with this dismissal

of the subject is that modelling conditional distributions alone is not always valid. In fact we can categorise at least three different types of covariate shift:

1. Independent covariate shift: $P_{train}(\mathbf{y}|\mathbf{x}) = P_{test}(\mathbf{y}|\mathbf{x})$, but $P_{train}(\mathbf{x}) \neq P_{test}(\mathbf{x})$.

2. Dependent prior probability change: $P_{train}(\mathbf{x}|\mathbf{y}) = P_{test}(\mathbf{x}|\mathbf{y})$, but $P_{train}(\mathbf{y}) \neq P_{test}(\mathbf{y})$.

3. Latent prior probability change: $P_{train}(\mathbf{x},\mathbf{y}|\mathbf{r}) = P_{test}(\mathbf{x},\mathbf{y}|\mathbf{r})$ for all values of some latent variable $\mathbf{r}$, but $P_{train}(\mathbf{r}) \neq P_{test}(\mathbf{r})$.

Let us presume that we are only interested in the quality of the conditional model $P_{test}(\mathbf{y}|\mathbf{x})$. Then Case 1 is the only one of the above where covariate shift will have no effect on modelling. Case 2 is the well known situation of class prior probability change and, for example, is considered in comparing the benefits of a naive Bayes model, which allows for class prior probability change, and discriminant models, which typically do not.

Case 3 involves a more general assumption, and arguably can be used to cover most situations of covariate shift, by incorporating any known structural characteristics of the problem into some latent variable $\mathbf{r}$. Change in the distribution of $\mathbf{x}$ points implicitly informs us about variation in the targets $\mathbf{y}$ via the shift in the latent variable $\mathbf{r}$, which is the causal factor for the change. The purpose of this paper is to provide a generative framework for analysis of covariate shift. The main advantages of this new formulation over previous approaches are

- It provides an explicit model for the changes occurring in situations of covariate shift, and hence the predictions that result from it.

- There is no need to presume the training and test distributions are known. Furthermore the test covariates are also used as part of the model estimation procedure, resulting in better predictions.

- Previous results, such as Importance Weighted Least Squares, are special cases of this method with explicit presumptions that can be relaxed to gain more general models. Hence this paper is a natural extension to the existing work.

- Utilising the test covariate distribution gives performance benefits over using the same model for training data alone.

- All the usual machinery for mixture of experts are available, and so this approach allows model selection and many natural extensions.

**Outline.** In Section 2, related work is discussed, before the problem is formally specified and a general model is derived in Section 3. A specific form of mixture regression model is formulated and an Expectation Maximisation solution is given in Section 3.1. The specific relationship to Importance Weighted Least Squares is discussed in Section 3.1.2. Test examples are given in section 4. The results and methods are discussed in Section 5.

## 2  Prior work

Covariate shift will be interpreted, in the context of this work, using mixture of regressor models, where the regression model is dependent on a latent class variable. Clustered regression models have been discussed widely [4, 18, 8, 16]. The benefits of the mixture of regressor approach for heterogeneous data was discussed in [17], but not formulated specifically for the problem of covariate shift. This paper establishes for the first time the relationship between the mixture of regressor model and the typical statistical results in the literature on covariate shift. The main differences of our approach from a standard mixture of regressor formalism is that we utilise the training and test distributions as part of the model and do not use only a conditional model, and we allow coupling of regressors across different mixture components. The main significance with regard to the literature on covariate shift is that we establish covariate shift within a general probabilistic modelling paradigm and hence extend the standard techniques to establish more general methods, which are also applicable when the training and test distributions are not explicitly given. The mixture of regressors form for $(\mathbf{x},\mathbf{y})$ used in this paper is a specific from of mixture of experts [10]. Hence hierarchical extensions are also possible in the form of [11].

The problem of sample selection bias is related to covariate shift. Sample selection bias has been discussed in [19], where they estimate the distribution determining the bias for a classification problem. The problem of sample selection bias differs from the case in this paper as here there is no fundamental requirement of distribution overlap between the training and test sets. First, each can have zero density in regions the other is non-zero. Second, the presumption is different: rather than there being a sample rejection process that characterised the difference between training and test sets, there is a sample production process that differs.

## 3 Framework for Covariate Shift

This paper follows most others in considering the restricted case of a single training and single test set. Each datum $\mathbf{x}$ is assumed to have been generated from one of a number of data sources using a mixture distribution corresponding to the source. The proportions of each of the sources varies across the training and test datasets. Hence, in the context of this paper, we understand covariate shift to be effected by a change in the contribution of different sources to the data.

The motivation of the framework in this paper is that there is a latent feature set upon which each dataset is dependent, and the the variations between the two datasets are dependent upon variation of the proportions, but not the form, of those latent features. This is characterised by presuming each data source is a member of one of two different sets. Each of the two sets of sources is also associated with a regression model. The two sets of sources have the following characteristics:

- Source set 1 corresponds to sources that may occur in the test data, and potentially also in the training data, and are associated with regression model $P_1(\mathbf{y}|\mathbf{x})$.

- Source set 2 corresponds to sources that occur only in the training data, and are associated with regression model $P_2(\mathbf{y}|\mathbf{x})$.

By taking this approach we note that we will be able to separate out effects that we expect to be only characteristics of the training data from effects that are common across training and test sets.

The full generative model for the observed data consists of the model for the training data $D$ and model for the test data $T$. The test data is just used to determine the nature of the covariate shift, and consists of only of the covariates $\mathbf{x}$, and not any targets $\mathbf{y}$. We emphasise that we do not presume to have seen the test data we wish to predict. Rather a prior model is built for the training and test data, and this is then conditioned on the information from the training data and the known covariates for the test data but not the unknown targets.

### 3.1 Mixture Regression for Covariate Shift

In this section the full model is introduced. This significantly extends the previous work on covariate shift, in that the model allows for unknown training and test distributions, and utilises a mixture model approach for representing the relationship between the two. In Section 3.1.2, we will show how the previous results on covariate shift are special cases of the general model. We will develop this formalism for any parametric form for the regressors $P(\mathbf{y}|\mathbf{x})$. In fact this restriction is mainly for ease of explanation, and the method can be used with non-parametric models too, and will be tested in the case of Gaussian process models[1].

The model takes the following form

- The distribution of the training data and test data are denoted $P_D$ and $P_T$ respectively, and are unknown in general.

- Source set 1 consists of $M$ mixture distributions, where mixture $t$ is denoted $P_{1t}(\mathbf{x})$. Each of the components is associated[2] with regression model $P_1(\mathbf{y}|\mathbf{x})$.

- Source set 2 consists of $M_2$ mixture distributions, where mixture $t$ is denoted $P_{2t}(\mathbf{x})$. Each of the components is associated with the regression model $P_2(\mathbf{y}|\mathbf{x})$.

- The training and test data distributions take the following form:

$$P_D(\mathbf{x}) = \sum_t \beta_1 \gamma_{1t}^D P_{1t}(\mathbf{x}) + \beta_2 \gamma_{2t}^D P_{2t}(\mathbf{x}) \text{ and } P_T(\mathbf{x}) = \gamma_{1t}^T P_{1t}(\mathbf{x}) \qquad (1)$$

Hence $\beta_1$ and $\beta_2$ are parameters for the proportions of the two source sets in the training data, $\gamma_{1t}^D$ are the relative proportions of each mixture from source set 1 in the training data, and $\gamma_{2t}^D$ are the relative proportions of each mixture from source set 2 in the training data. Finally $\gamma_{1t}^T$ are the proportions of each mixture from source set 1 in the test data. All these parameters are presumed unknown. At some points in the paper it will be presumed the mixtures are Gaussian, when the form $N(\mathbf{x}; \mathbf{m}, \mathbf{K})$ will be used to denote the Gaussian distribution function of $\mathbf{x}$, with mean $\mathbf{m}$ and covariance $\mathbf{K}$.

For a parametric model, with the collection of mixture parameters denoted by $\boldsymbol{\Omega}$, the collection of regression parameters denoted by $\boldsymbol{\Theta}$, and the mixing proportions, $\boldsymbol{\gamma}$ and $\boldsymbol{\beta}$ we have the full probabilistic model

$$P(\{i^\mu, \mathbf{y}^\mu, \mathbf{x}^\mu | \mu \in D\}, \{i^\nu, \mathbf{x}^\nu | \nu \in T\} | \boldsymbol{\beta}, \boldsymbol{\Theta}, \boldsymbol{\Omega}) =$$
$$\prod_{\mu \in D} P(s^\mu | \boldsymbol{\beta}) P(t^\mu | \boldsymbol{\gamma}, s^\mu) P_{s^\mu t^\mu}(\mathbf{x}^\mu | \Omega_{t^\mu}) P_{s^\mu}(\mathbf{y}^\mu | \mathbf{x}^\mu, \boldsymbol{\Theta}) \prod_{\nu \in T} P(t^\nu | \boldsymbol{\gamma}) P_{1t^\mu}(\mathbf{x}^\nu | \boldsymbol{\Omega}). \quad (2)$$

where $s^\mu$ denotes the source set used to generate the data point $\mu$, and $t^\mu$ denotes the particular mixture from that source set used to generate the data point $\mu$. In words, this says that the model for the training dataset involves sampling the particular source set $s^\mu$, then the mixture component $t^\mu$ from that particular source set. Given these we then sample an $\mathbf{x}^\mu$ from the relevant mixture and a $\mathbf{y}^\mu$ conditionally on $\mathbf{x}^\mu$ from the relevant regressor. The same procedure is followed for the test set, except now there is only one source set to consider.

### 3.1.1 EM algorithm

A maximum likelihood solution for the parameters $(\boldsymbol{\beta}, \boldsymbol{\gamma}, \boldsymbol{\Theta}, \boldsymbol{\Omega})$ can be obtained for this model (given the training data and test covariate) using Expectation Maximisation (EM) [3]. The derivations are standard EM calculations (see e.g. [2]), and hence are not reiterated here. Denote the responsibility of mixture $i$ for data point $\mu$ by $\alpha_i^\mu$. Then the application of EM involves maximisation of

$$\log P(\{\mathbf{y}^\mu, \mathbf{x}^\mu | \mu \in D\}, \{\mathbf{x}^\nu | \nu \in T\} | \boldsymbol{\beta}, \boldsymbol{\gamma}, \boldsymbol{\Theta}, \boldsymbol{\Omega}) \qquad (3)$$

with respect to the parameters through iteration of E and M steps. The E-step update uses current parameter values to compute the responsibility (denoted by $\alpha$s) of each mixture $1t$ and $2t$ for each data point $\mu$ in the training set and each data point $\nu$ in the test set using

$$\alpha_{st}^\mu = \frac{\beta_s \gamma_{st}^D P_{st}(\mathbf{x}^\mu | \boldsymbol{\Omega}) P_s(\mathbf{y}^\mu | \mathbf{x}^\mu, \boldsymbol{\Theta})}{\sum_{s,t} \beta_s \gamma_{st}^D P_{st}(\mathbf{x}^\mu | \boldsymbol{\Omega}) P_s(\mathbf{y}^\mu | \mathbf{x}^\mu, \boldsymbol{\Theta})} \text{ and } \alpha_{1t}^\nu = \frac{\gamma_{1t}^T P_{1t}(\mathbf{x}^\mu | \boldsymbol{\Omega})}{\sum_t \gamma_{1t}^T P_{1t}(\mathbf{x}^\mu | \boldsymbol{\Omega})}. \qquad (4)$$

We set $\alpha_{2t}^\mu = 0$ for $\nu \in T$, as none of these mixtures are represented in the test set. The parameters of the mixture model distributions are then updated with the usual M steps for the relevant mixture component, and the regression parameters are updated using maximum responsibility-weighted likelihood. When each mixture component is a Gaussian of the form $N(\mathbf{x}; \mathbf{m}_{st}, \mathbf{K}_{st})$, when we have a Gaussian regression error term, and denoting the (vector of) regression functions by $\mathbf{f}_s$ for each source set $s$, these update rules are:

$$\mathbf{m}_{st} = \frac{\sum_{\mu \in (D,T)} \alpha_{st}^\mu \mathbf{x}^\mu}{\sum_{\mu \in D,T} \alpha_{st}^\mu}, \mathbf{K}_{st} = \frac{\sum_{\mu \in (D,T)} \alpha_{st}^\mu (\mathbf{x}^\mu - \mathbf{m}_{st})(\mathbf{x}^\mu - \mathbf{m}_{st})^T}{\sum_{\mu \in D,T} \alpha_{st}^\mu} \qquad (5)$$

$$\beta_s = \frac{1}{|D|} \sum_{\mu \in D,t} \alpha_{st}^\mu, \gamma_{st}^D = \frac{1}{|D|} \sum_{\mu \in D} \frac{\alpha_{st}^\mu}{\beta_s}, \gamma_{1t}^T = \frac{1}{|T|} \sum_{\nu \in T} \alpha_{1t}^\nu \qquad (6)$$

$$\mathbf{f}_s = \arg\min_{\mathbf{f}_s} \left[ \sum_{\mu,t} \alpha_{st}^\mu (\mathbf{f}_s(\mathbf{x}^\mu) - \mathbf{y}^\mu)^T (\mathbf{f}_s(\mathbf{x}^\mu) - \mathbf{y}^\mu) \right] \qquad (7)$$

Given the learnt model, inference is straightforward. The test data is associated with a single regression model $P_1(\mathbf{y}|\mathbf{x})$, and so the predictive distribution for the test set is the learnt predictor $P_1(\mathbf{y}|\mathbf{x}_i)$ for each point $\mathbf{x}_i$ in the test set.

### 3.1.2 Importance Weighted Least Squares

Previous results in modelling covariate shift can be obtained as special cases of the general approach taken in this paper. Suppose we make the assumptions that $P_D$ and $P_T$ are known, and that the source set 1 contains just the one component, which must be $P_T$ by definition. Suppose also that the two regressors have a large and identical variance $\Sigma$. In this simple case, we do not need to know the actual test points (in this framework these are only used to infer the test distribution, which is assumed given here). The M step update only involves update to the regressor. For the E step we use the approximation $P(\mathbf{y}^\mu|\mathbf{x}^\mu, \Theta_1) \approx P(\mathbf{y}^\mu|\mathbf{x}^\mu, \Theta_2)$, which becomes asymptotically true in the case of infinite variance $\Sigma$. The resulting $E$ and $M$ steps are

$$\alpha^\mu \approx \frac{P_T(\mathbf{x}^\mu)\beta_1}{P_D(\mathbf{x}^\mu)} \text{ and } \mathbf{f}_1 = \arg\min_{\mathbf{f}_1} \left[ \sum_\mu \alpha^\mu (\mathbf{f}_1(\mathbf{x}^\mu) - \mathbf{y}^\mu)^T (\mathbf{f}_1(\mathbf{x}^\mu) - \mathbf{y}^\mu) \right] \tag{8}$$

where we note that $\beta_1$ is a common constant and can be dropped from the calculations. Hence we never need to learn $\beta_1$ or the parameters associated with mixture 2 in this procedure. Also no iterative EM procedure is needed as the E step is independent of the M step results. Hence this is a one shot process. This is the Importance Weighted Least Squares estimator for covariate shift [14]. A simple extension of this model will allow the large variance assumption to be relaxed, so the model can use the regressor information for computing responsibilities.

## 4 Examples

### 4.1 Generated Test Data

We demonstrate the mixture of regressors approach to covariate shift (MRCS) on generated test data: a one dimensional regression problem with two sources each corresponding to different linear regressors. Regression performance for MRCS with Gaussian mixtures and linear regressors is compared with three other cases. The first is an importance weighted least squares estimator (IWLS) given the best mixture model fit for the data, corresponding to the current standard for modelling covariate shift. The second uses a mixture of regressors model that ignores the form of the test data, but chooses the regressor corresponding to the mixtures which best match the test data distribution using a KL divergence measure (MRKL). This corresponds to recognising that covariate shift can happen, but ignoring the nature of the test distribution in the modelling process, and trying to choose the best of the two regressors. The third case is where the mixture of regressors is used simply as a standard regression model, ignoring the possibility of covariate shift (MRREG).

The generative procedure for each of the 100 test datasets involves generating random parameter values for between 1 and 3 mixtures for each of two linear regressors. Test and training datasets of 200 data points each are generated from these mixtures and regression models, using different mixing proportions in each case. The various approaches were run 8 times with different random starting parameters for all methods. 80 iterations of EM were used. A fixed number of iterations was chosen to allow reasonable comparison. Analysis was done for fixed model sizes and for model choice using a Bayesian Information Criterion (BIC). Even though the regularity conditions for BIC do not hold for mixture models, it has been shown that BIC is consistent in this case [12]. It has also been shown to be a good choice on practical grounds [6].

The results of these tests show the significant benefits of explicit recognition of covariate shift over straight regression even compared with the use of the same mixture of regressors model, but without reference to the test distribution. It also shows benefits of the approach of this paper over the current state of the art for modelling covariate shift. Table 1 gives the result of these approaches for various fixed choices of numbers of mixtures associated with each regressor. Independent of the use of any model order choice, the Mixture of Regressors for Covariate Shift (MRCS) performs better than the other approaches. Table 1 also gives the results when the Bayesian Information Criterion is used for selecting the number of mixtures. Again MRCS performs best, and consistently gives better performance on the test data for more than 70 percent of the test cases.

To illustrate the difference between the methods, Figure 1 plots the results of training a MRCS model on some one dimensional data using a regularised cubic regressor. The fit to the test data is also shown. Once again this is compared with IWLS and MRKL. It can be seen that both IWLS and MRKL fail to untangle the regressors associated with the overlapping central clusters in the training data and hence perform badly in that region of the test data.

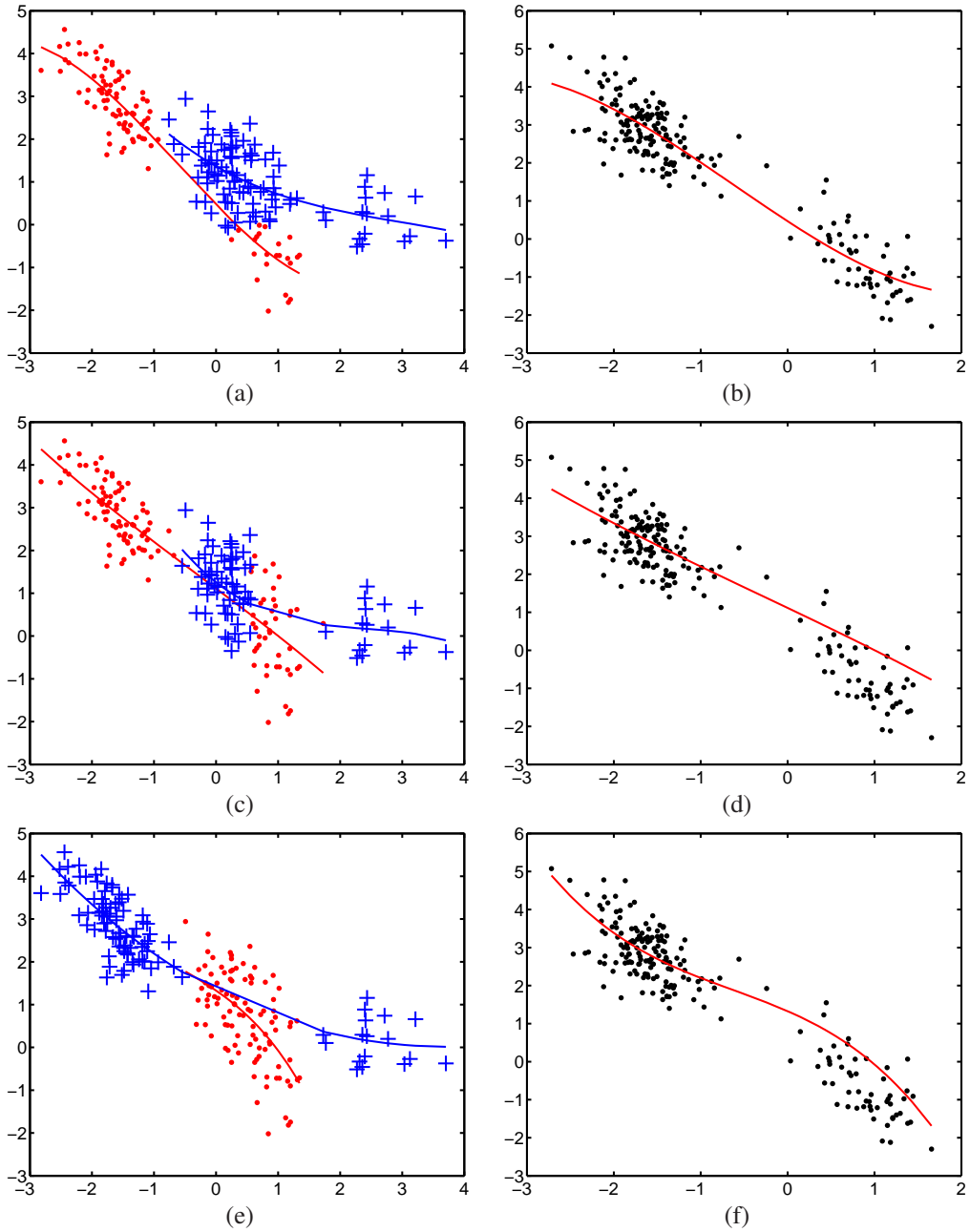

Figure 1: Nonlinear regression using covariate shift. (a),(c),(e) Training set fit and (b),(d),(f) test data with predictions for MRCS (top), IWLS (middle) and MRKL (bottom) respectively. In (a),(c),(e), the '.' data labels the points for which the test regressor has greater responsibility, and the '+' data labels points for which the training only regressor has greater responsibility.

Table 1: Average mean square error over all 100 datasets for each choice of fixed model mixture size. The actual number of mixtures in the data varies. MRCS - mixture of regressors for covariate shift, IWLS - Importance weighted least squares, MRKL - Mixture of regressors, evaluated on regressor with best fit to test distribution. MRREG - Mixture of regressors as a standard regression model, ignoring covariate shift. The sixth row gives the average mean square error over all 100 datasets, with number of mixtures chosen using a Bayesian information criterion for each case, and the last row gives the proportion of times MRCS performs better than the other cases for a BIC choice of model. PValues: If two of the approaches were equivalent performers, empirically better performance for 70/100 or more cases would only occur on less than $1 \times 10^{-4}$ such trials.

|            | MRCS   | IWLS   | MRKL   | MRREG  |
|------------|--------|--------|--------|--------|
| 1 Mixture  | 0.588  | 0.797  | 3.274  | 0.890  |
| 2 Mixtures | 0.536  | 0.804  | 2.673  | 0.881  |
| 3 Mixtures | 0.601  | 0.831  | 3.390  | 0.887  |
| 4 Mixtures | 0.623  | 0.817  | 2.823  | 0.894  |
| 5 Mixtures | 0.612  | 0.837  | 2.817  | 0.898  |
| BIC Choice | 0.6100 | 0.7990 | 2.8638 | 0.8813 |
| MCRS better | -     | 77/100 | 72/100 | 84/100 |

## 4.2 Auto-Mpg Test

It is useful to see that the approach does indeed make a noticeable difference on data that takes the appropriate prior form, but that says nothing about how appropriate that prior is for real problems. Here we demonstrate the method on the `auto-mpg` problem from the UCI dataset. This provides a natural scenario for demonstrating covariate shift. The auto-mpg data can be found at `http://www.ics.uci.edu/~mlearn/MLSummary.html` and involves the problem of predicting the city cycle fuel consumption of cars. One of the attributes is a class label dictating the origin of a particular car. To demonstrate covariate shift we can consider the prediction task trained on cars from one place of origin and tested on cars from another place of origin. Here we consider predicting the fuel consumption (attribute 1) using the four continuous attributes. We train the model using data on cars from origin 1, and test on cars from origin 2 and origin 3. We use the same test algorithms as the previous section, but now using a Gaussian process regressors for each regression function. The results of running this are in Table 2. The Gaussian process hyper-parameters were optimised separately for each case. These are results obtained using a Bayesian Information Criterion for selecting the number of mixtures between 1 and 14 for each of the cases. We obtain similar results if we compare methods with various fixed numbers of mixtures. Critically, we note that all covariate shift methods performed better than a straight Gaussian Process predictor in this situation. The mixture of Gaussian processes did not perform as well as the methods which explicitly recognised the covariate shift, although interestingly did perform better than a straight Gaussian process predictor. Again the MRCS performed better overall.

## 5  Discussion

This paper establishes that explicit generative modelling of covariate shift can bring improvements over conditional regression models, or over standard covariate shift methods that ignore the dependent data in the modelling process. The method is also better than using an *identical* mixture of regressors model for the training data alone, as it utilises the positions of the independent test points to help refine the mixture locations and the separation of regressors.

We expect significant improvements can be made with a fully Bayesian treatment of the parameters. This framework is currently being extended to the case of multiple training and test datasets using a fully Bayesian scheme, and will be the subject of future work. In this setting we have a Topic model,

Table 2: Tests of methods on the `auto-mpg` dataset. These are the (standardised) mean squared errors for each approach. GP denotes the use of Gaussian Process regression for prediciton. Orgin 2, and Origin 3 denote the two different car origins used to test the model.

|          | GP    | MRCS  | IWLS  | MRKL   | MRREG  |
|----------|-------|-------|-------|--------|--------|
| Origin 2 | 1.192 | 0.600 | 0.700 | 1.2243 | 0.7397 |
| Origin 3 | 0.898 | 0.568 | 0.691 | 1.3862 | 0.706  |

similar to Latent Dirichlet Allocation, where each dataset is built from a number of contributing regression components, where each component is expressed in different proportions in each dataset. The model and tests of this paper show that this multiple dataset extension could well be fruitful.

## 6 Conclusion

In this paper a novel approach to the problem of covariate shift has been developed that is demonstrably better than state of the art regression approaches, and better than the current standard for covariate shift. These have been tested on both generated data, and on a real problem of covariate shift, derived from a standard UCI dataset. Importance Weighted Least Squares is shown to be a special case. Specifically we provide explicit modelling of the covariate shift process by assuming a shift in the proportions of a number of latent components. A mixture of regressors model is used for this purpose, but it differs from standard mixture of regressors by allowing sharing of the regression functions between mixture components and explicitly including a model for the test set as part of the process.

## Footnotes

[1]The primary restriction is than we need to be able to compute standard EM responsibilites for a given regressor, and hence for Gaussian processes a variational approximation is needed to do this.

[2]If a component $i$ is associated with a regression model $j$, this means that any datum $\mathbf{x}$ generated from the mixture component $i$, will also have a corresponding $\mathbf{y}$ generated from the associated regression model $P_j(\mathbf{y}|\mathbf{x})$.

## References

[1] P. Baldi, S. Brumak, and G. A. Stolovitzky. *Bioinformatics: The Machine Learning Approach*. MIT Press, Cambridge, 1998.

[2] C. M. Bishop. *Neural Networks for Pattern Recognition*. Oxford University Press, 1995.

[3] A. Dempster, N. Laird, and D. Rubin. Maximum likelihood from incomplete data via the EM algorithm. *Journal of the Royal Statistical Society*, 39:1–38, 1977.

[4] W.S. DeSarbo and W.L. Cron. A maximum likelihood methodology for clusterwise linear regression. *Journal of Classification*, 5:249–282, 1988.

[5] R. O. Duda, P. E. Hart, and D. G. Stork. *Pattern Classification*. Wiley Interscience, 2001.

[6] C. Fraley and A.E. Raftery. How many clusters? Which clustering method? Answers via model-based cluster analysis. *Computer Journal*, 41:578–588, 1998.

[7] J. J. Heckman. Sample selection bias as a specification error. *Econometrica*, 47:153–162, 1979.

[8] C. Hennig. Identifiability of models for clusterwise linear regressions. *Journal of Classification*, 17:273–296, 2000.

[9] R. Herbrich. *Learning Kernel Classifiers*. MIT Press, 2002.

[10] R.A. Jacobs, M.I. Jordan, S.J. Nowlan, and G.E. Hinton. Adaptive mixtures of local experts. *Neural Computation*, 3:79–87, 1991.

[11] M. I. Jordan and R. A. Jacobs. Hierarchical mixtures of experts and the EM algorithm. *Neural Computation*, 6:181–214, 1994.

[12] C. Keribin. Consistent estimation of the order of mixture models. Technical report, Université d'Evry-Val d'Essonne, Laboratoire Analyse et Probabilité, 1997.

[13] C.R. Shelton. *Importance Sampling for Reinforcement Learning with multiple Objectives*. PhD thesis, Massachusetts Institute of Technology, 2001.

[14] H. Shimodaira. Improving predictive inference under covariate shift by weighting the log-likelihood function. *Journal of Statistical Planning and Inference*, 90:227–244, 2000.

[15] M. Sugiyama and K. -R. Müller. Input-dependent estimation of generalisation error under covariate shift. *Statistics and Decisions*, 23:249–279, 2005.

[16] H.G. Sung. *Gaussian Mixture Regression and Classification*. PhD thesis, Rice University, 2004.

[17] J.K. Vermunt. A general non-parametric approach to unobserved heterogeneity in the analysis of event history data. In J. Hagenaars and A. McCutcheon, editors, *Applied Latent Class Models*. Cambridge University Press, 2002.

[18] M. Wedel and W.S. DeSarbo. A mixture likelihood approach for generalised linear models. *Journal of Classification*, 12:21–55, 1995.

[19] B. Zadrozny. Learning and evaluating classifiers under sample selection bias. In *Proceedings of ICML*, 2004.
